# Hierarchical Dirichlet Processes with Random Effects

**Seyoung Kim**
Department of Computer Science
University of California, Irvine
Irvine, CA 92697-3435
sykim@ics.uci.edu

**Padhraic Smyth**
Department of Computer Science
University of California, Irvine
Irvine, CA 92697-3435
smyth@ics.uci.edu

## Abstract

Data sets involving multiple groups with shared characteristics frequently arise in practice. In this paper we extend hierarchical Dirichlet processes to model such data. Each group is assumed to be generated from a template mixture model with group level variability in both the mixing proportions and the component parameters. Variabilities in mixing proportions across groups are handled using hierarchical Dirichlet processes, also allowing for automatic determination of the number of components. In addition, each group is allowed to have its own component parameters coming from a prior described by a template mixture model. This group-level variability in the component parameters is handled using a random effects model. We present a Markov Chain Monte Carlo (MCMC) sampling algorithm to estimate model parameters and demonstrate the method by applying it to the problem of modeling spatial brain activation patterns across multiple images collected via functional magnetic resonance imaging (fMRI).

## 1 Introduction

Hierarchical Dirichlet processes (DPs) (Teh et al., 2006) provide a flexible framework for probabilistic modeling when data are observed in a grouped fashion and each group can be thought of as being generated from a mixture model. In the hierarchical DPs all of, or a subset of, the mixture components are shared by different groups and the number of such components are inferred from the data using a DP prior. Variability across groups is modeled by allowing different mixing proportions for different groups.

In this paper we focus on the problem of modeling systematic variation in the shared mixture component parameters and not just in the mixing proportions. We will use the problem of modeling spatial fMRI activation across multiple brain images as a motivating application, where the images are obtained from one or more subjects performing the same cognitive tasks. Figure 1 illustrates the basic idea of our proposed model. We assume that there is an unknown true template for mixture component parameters, and that the mixture components for each group are noisy realizations of the template components. For our application, groups and data points correspond to images and pixels. Given grouped data (e.g., a set of images) we are interested in learning both the overall template model and the random variation relative to the template for each group. For the fMRI application, we model the images as mixtures of activation patterns, assigning a mixture component to each spatial activation cluster in an image. As shown in Figure 1 our goal is to extract activation patterns that are common across multiple images, while allowing for variation in fMRI signal intensity and activation location in individual images. In our proposed approach, the amount of variation (called random effects) from the overall true component parameters is modeled as coming from a prior distribution on group-level component parameters (Gelman et al. 2004). By combining hierarchical DPs with a random effects model we let both mixing proportions and mixture component parameters adapt to the data in each group. Although we focus on image data in this paper, the proposed

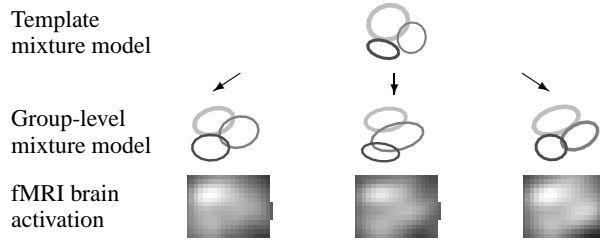

Template mixture model

Group-level mixture model

fMRI brain activation

Figure 1: Illustration of group level variations from the template model.

| Model | Group-level mixture components |
|---|---|
| Hierarchical DPs | $\theta_a \times m_a, \ \theta_b \times m_b$ |
| Transformed DPs | $\theta_a + \Delta_{a1}, \ \ldots, \ \theta_a + \Delta_{am_a}, \ \theta_b + \Delta_{b1}, \ \ldots, \ \theta_b + \Delta_{bm_b}$ |
| Hierarchical DPs with random effects | $(\theta_a + \Delta_a) \times m_a, \ (\theta_b + \Delta_b) \times m_b$ |

Table 1: Group-level mixture component parameters for hierarchical DPs, transformed DPs, and hierarchical DPs with random effects as proposed in this paper.

approach is applicable to more general problems of modeling group-level random variation with mixture models.

Hierarchical DPs and transformed DPs (Sudderth et al., 2005) both address a similar problem of modeling groups of data using mixture models with mixture components shared across groups. Table 1 compares the basic ideas underlying these two models with the model we propose in this paper. Given a template mixture of two components with parameters $\theta_a$ and $\theta_b$, in hierarchical DPs a mixture model for each group can have $m_a$ and $m_b$ exact copies (commonly known as tables in the Chinese restaurant process representation) of each of the two components in the template—thus, there is no notion of random variation in component parameters across groups. In transformed DPs, each of the copies of $\theta_a$ and $\theta_b$ receives a transformation parameter $\Delta_{a1}, \ldots, \Delta_{am_a}$ and $\Delta_{b1}, \ldots, \Delta_{bm_b}$. This is not suitable for modeling the type of group variation illustrated in Figure 1 because there is no direct way to enforce $\Delta_{a1} = \ldots = \Delta_{am_a}$ and $\Delta_{b1} = \ldots = \Delta_{bm_b}$ to obtain $\Delta_a$ and $\Delta_b$ as used in our proposed model.

In this general context the model we propose here can be viewed as being closely related to both hierarchical DPs and transformed DPs, but having application to quite different types of problems in practice, e.g., as an intermediate between the highly constrained variation allowed by the hierarchical DP and the relatively unconstrained variation present in the computer vision scenes to which the transformed DP has been applied (Sudderth et al, 2005).

From an applications viewpoint the use of DPs for modeling multiple fMRI brain images is novel and shows considerable promise as a new tool for analyzing such data. The majority of existing statistical work on fMRI analysis is based on voxel-by-voxel hypothesis testing, with relatively little work on modeling of the spatial aspect of the problem. One exception is the approach of Penny and Friston (2003) who proposed a probabilistic mixture model for spatial activation modeling and demonstrated its advantages over voxel-wise analysis. The application of our proposed model to fMRI data can be viewed as a generalization of Penny and Friston's work in three different aspects by (a) allowing for analysis of multiple images rather than a single image (b) learning common activation clusters and systematic variation in activation across these images, and (c) automatically learning the number of components in the model in a data-driven fashion.

## 2 Models

### 2.1 Dirichlet process mixture models

A Dirichlet process $DP(\alpha_0, G)$ with a concentration parameter $\alpha_0 > 0$ and a base measure $G$ can be used as a nonparametric prior distribution on mixing proportion parameters in a mixture model when the number of components is unknown *a priori* (Rasmussen, 2000). The generative process for a mixture of Gaussian distributions with component mean $\mu_k$ and DP prior $DP(\alpha_0, G)$ can be

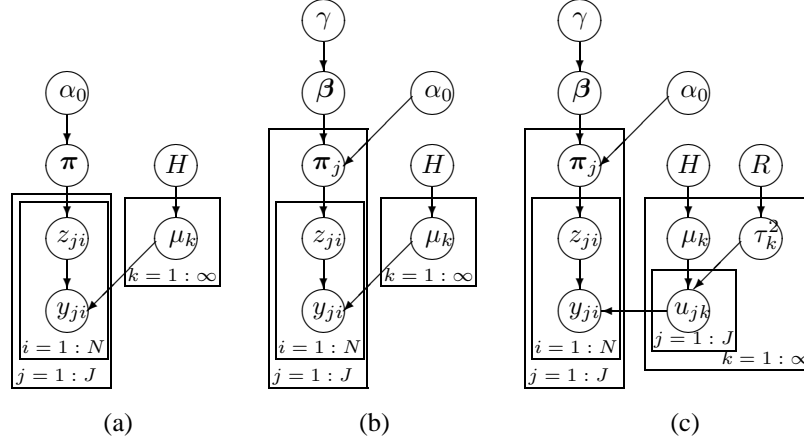

<div align="center">(a)        (b)        (c)</div>

Figure 2: Plate diagrams for (a) DP mixtures, (b) hierarchical DPs and (c) hierarchical DPs with random effects.

written, using a stick breaking construction (Sethuraman, 2004), as:

$$\boldsymbol{\pi}|\alpha_0 \sim \text{Stick}(\alpha_0), \quad \mu_k|G \sim \mathcal{N}_G(\mu_0, \psi_0^2), \quad z_i|\boldsymbol{\pi} \sim \boldsymbol{\pi}, \quad y_i|z_i, (\mu_k)_{k=1}^{\infty}, \sigma^2 \sim \mathcal{N}(\mu_{z_i}, \sigma^2),$$

where $y_i, i = 1, \ldots, N$ are observed data and $z_i$ is a component label for $y_i$. It can be shown that the labels $z_i$'s have the following clustering property:

$$z_i|z_1, \ldots, z_{(i-1)}, \alpha_0 \sim \sum_{k=1}^{K} \frac{n_k^{-i}}{i-1+\alpha_0}\delta_k + \frac{\alpha_0}{i-1+\alpha_0}\delta_{k_{\text{new}}},$$

where $n_k^{-i}$ represents the number of $z_{i'}, i' \neq i$, assigned to component $k$. The probability that $z_i$ is assigned to a new component is proportional to $\alpha_0$. Note that the component with more observations already assigned to it has a higher probability to attract the next observation.

## 2.2 Hierarchical Dirichlet processes

When multiple groups of data are present and each group can be modeled as a mixture it is often useful to let different groups share mixture components. In hierarchical DPs (Teh et al., 2006) components are shared by different groups with varying mixing proportions for each group, and the number of components in the model can be inferred from data.

Let $y_{ji}$ be the $i$th data point ($i = 1, \ldots, N$) in group $j$ ($j = 1, \ldots, J$), $\boldsymbol{\beta}$ the global mixing proportions, $\boldsymbol{\pi}_j$ the mixing proportions for group $j$, and $\alpha_0, \gamma, H$ are the hyperparameters for the DP. Then, the hierarchical DP can be written as follows, using a stick breaking construction:

$$\begin{aligned} \boldsymbol{\beta}|\gamma \sim \text{Stick}(\gamma), \qquad \boldsymbol{\pi}_j|\alpha_0, \boldsymbol{\beta} \sim \text{DP}(\alpha_0, \boldsymbol{\beta}), \qquad z_{ji}|\boldsymbol{\pi}_j \sim \boldsymbol{\pi}_j, \\ \mu_k|H \sim \mathcal{N}_H(\mu_0, \psi_0^2), \qquad y_{ji}|z_{ji}, (\mu_k)_{k=1}^{\infty}, \sigma^2 \sim \mathcal{N}(\mu_{z_{ji}}, \sigma^2), \end{aligned} \tag{1}$$

The plate diagram in Figure 2(b) illustrates the generative process of this model. Mixture components described by the $\mu_k$'s can be shared across the $J$ groups.

The hierarchical DP has clustering properties similar to that for DP mixtures, i.e.,

$$p(h_{ji}|\mathbf{h}_{-ji}, \alpha_0) \sim \sum_{t=1}^{T_j} \frac{n_{jt}^{-i}}{n_j - 1 + \alpha_0}\delta_t + \frac{\alpha_0}{n_j - 1 + \alpha_0}\delta_{t_{\text{new}}} \tag{2}$$

$$p(l_{jt}|\mathbf{l}_{-jt}, \gamma) \sim \sum_{k=1}^{K} \frac{m_k^{-t}}{\sum m_u - 1 + \gamma}\delta_k + \frac{\gamma}{\sum m_u - 1 + \gamma}\delta_{k_{\text{new}}}, \tag{3}$$

where $h_{ji}$ represents the mapping of each data item $y_{ji}$ to one of $T_j$ clusters within group $j$ and $l_{jt}$ maps the $t$th local cluster in group $j$ to one of $K$ global clusters shared by all of the $J$ groups.

The probability that a new local cluster is generated within group $j$ is proportional to $\alpha_0$. This new cluster is generated according to Equation (3). Notice that more than one local cluster in group $j$ can be linked to the same global cluster. It is the assignment of data items to $K$ global clusters via local cluster labels that is typically of interest.

## 3 Hierarchical Dirichlet processes with random effects

We now propose an extension of the standard hierarchical DP to a version that includes random effects. We first develop our model for the case of Gaussian density components, and later in the paper apply this model to the specific problem of modeling activation patterns in fMRI brain images.

We take $\mu_k|H \sim \mathcal{N}_H(\mu_0, \psi_0^2)$ and $y_{ji}|z_{ji}, (\mu_k)_{k=1}^\infty, \sigma^2 \sim \mathcal{N}(\mu_{z_{ji}}, \sigma^2)$ in Equation (1) and add random effects as follows:

$$\mu_k|H \sim \mathcal{N}_H(\mu_0, \psi_0^2), \quad \tau_k^2|R \sim \text{Inv-}\chi_R^2(v_0, s_0^2),$$
$$u_{jk}|\mu_k, \tau_k^2 \sim \mathcal{N}(\mu_k, \tau_k^2), \quad y_{ji}|z_{ji}, (u_{jk})_{k=1}^\infty \sim \mathcal{N}(u_{jz_{ji}}, \sigma^2). \tag{4}$$

Each group $j$ has its own component mean $u_{jk}$ for the $k$th component and these group-level parameters come from a common prior distribution $\mathcal{N}(\mu_k, \tau_k^2)$. Thus, $\mu_k$ can be viewed as a template, and $u_{jk}$ as a noisy observation of the template for group $j$ with variance $\tau_k^2$. The random effects parameters $u_{jk}$ are generated once per group and shared by local clusters in group $j$ that are assigned to the same global cluster $k$.

For inference we use an MCMC sampling scheme that is based on the clustering property given in Equations (2) and (3). In each iteration we sample labels $\mathbf{h} = \{h_{ji} \text{ for all } j, i\}$, $\mathbf{l} = \{l_{jt} \text{ for all } j, t\}$ and component parameters $\boldsymbol{\mu} = \{\mu_k \text{ for all } k\}$, $\boldsymbol{\tau}^2 = \{\tau_k^2 \text{ for all } k\}$, $\mathbf{u} = \{u_{jk} \text{ for all } k, j\}$ alternately.

We sample $t_{ji}$'s using the following conditional distribution:

$$p(h_{ji} = t|\mathbf{h}_{-ji}, \mathbf{u}, \boldsymbol{\mu}, \boldsymbol{\tau}^2, \mathbf{y}) \propto \begin{cases} n_{-jt} p(y_{ji}|u_{jk}, \sigma^2) & \text{if } t \text{ was used} \\ \alpha_0 p(y_{ji}|\mathbf{h}_{-ji}\mathbf{u}, \boldsymbol{\mu}, \boldsymbol{\tau}^2, \gamma) & \text{if } t = t_{\text{new}}, \end{cases}$$

where

$$p(y_{ji}|\mathbf{h}_{-ji}\mathbf{u}, \boldsymbol{\mu}, \boldsymbol{\tau}, \gamma) = \sum_{k \in \mathbf{A}} \frac{m_k}{\sum_k m_k + \gamma} p(y_{ji}|u_{jk}) \tag{5a}$$

$$+ \sum_{k \in \mathbf{B}} \frac{m_k}{\sum_k m_k + \gamma} \int p(y_{ji}|u_{jk}) p(u_{jk}|\mu_k, \tau_k^2) \mathrm{d}u_{jk} \tag{5b}$$

$$+ \frac{\gamma}{\sum_k m_k + \gamma} \int \int \int p(y_{ji}|u_{jk}) p(u_{jk}|\mu_k, \tau_k^2) \mathcal{N}_H(\mu_0, \psi_0^2) \text{Inv-}\chi_R^2(v_0, s_0^2) \mathrm{d}u_{jk} \mathrm{d}\mu_k \mathrm{d}\tau_k^2. \tag{5c}$$

In Equation (5a) the summation is over components in $\mathbf{A} = \{k|\text{ some } h_{ji'} \text{ for } i' \neq i \text{ is assigned to } k\}$, representing global clusters that already have some local clusters in group $j$ assigned to them. In this case, since $u_{jk}$ is already known, we can simply compute the likelihood $p(y_{ji}|u_{jk})$. In Equation (5b) the summation is over $\mathbf{B} = \{k|\text{ no } h_{ji'} \text{ for } i' \neq i \text{ is assigned to } k\}$ representing global clusters that have not yet been assigned in group $j$. For conjugate priors we can integrate over the unknown random effects parameter $u_{jk}$ to compute the likelihood using $\mathcal{N}(y_{ji}|\mu_k, \tau_k^2 + \sigma^2)$ and sample $u_{jk}$ from the posterior distribution $p(u_{jk}|\mu_k, \tau_k^2, y_{ji})$. Equation (5c) models the case where a new global component gets generated. The integral cannot be evaluated analytically, so we approximate the integral by sampling new values for $\mu_k$, $\tau_k^2$, and $u_{jk}$ from prior distributions and evaluating $p(y_{ji}|u_{jk})$ given these new values for the parameters (Neal, 1998).

Samples for $l_{jt}$'s can be obtained from the conditional distribution given as

$$p(l_{jt} = k|\mathbf{l}_{-jt}, \mathbf{u}, \boldsymbol{\mu}, \boldsymbol{\tau}^2, \mathbf{y}) \propto \begin{cases} m_{-jt} \prod_{i:h_{ji}=t} p(y_{ji}|u_{jk}, \sigma^2) \\ \qquad \text{if } k \text{ was used in group } j \\ m_{-jt} \int \prod_{i:h_{ji}=t} p(y_{ji}|u_{jk}, \sigma^2) p(u_{jk}|\mu_k, \tau_k^2) \mathrm{d}u_{jk} \\ \qquad \text{if } k \text{ is new in group } j \\ \gamma \int \int \int \prod_{i:h_{ji}=t} p(y_{ji}|u_{jk}) p(u_{jk}|\mu_k, \tau_k^2) \\ \qquad \mathcal{N}_H(\mu_0, \psi_0^2) \text{Inv-}\chi_R^2(v_0, s_0^2) \; \mathrm{d}u_{jk} \mathrm{d}\mu_k \mathrm{d}\tau_k \\ \qquad \text{if } k \text{ is a new component.} \end{cases} \tag{6}$$

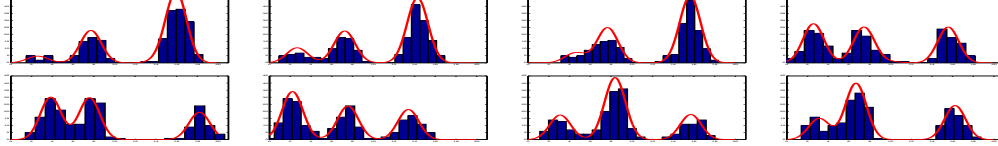

Figure 3: Histogram for simulated data with mixture density estimates overlaid.

As in the sampling of $h_{ji}$, if $k$ is new in group $j$ we can evaluate the integral analytically and sample $u_{jk}$ from the posterior distribution. If $k$ is a new component we approximate the integral by sampling new values for $\mu_k$, $\tau_k^2$, and $u_{jk}$ from the prior and evaluating the likelihood.

Given $\mathbf{h}$ and $\mathbf{l}$ we can update the component parameters $\boldsymbol{\mu}$, $\boldsymbol{\tau}$ and $\mathbf{u}$ using standard Gibbs sampling for a normal hierarchical model (Gelman et al., 2006). In practice, this Markov chain can mix poorly and get stuck in local maxima where the labels for two group-level components are swapped relative to the same two components in the template. To address this problem and restore the correct correspondence between template components and group-level components we propose a move that swaps the labels for two group-level components at the end of each sampling iteration and accepts the move based on a Metropolis-Hastings acceptance rule.

To illustrate the proposed model we simulated data from a mixture of one-dimensional Gaussian densities with known parameters and tested if the sampling algorithm can recover the parameters from the data. From a template mixture model with three mixture components we generated 10 group-level mixture models by adding random effects in the form of mean-shifts to the template means, sampled from $\mathcal{N}(0, 1)$. Using varying mixing proportions for each group we generated 200 samples from each of the 10 mixture models. Histograms for the samples in eight groups are shown in Figure 3(a). The estimated models after 1000 iterations of the MCMC algorithm are overlaid. We can see that the sampling algorithm was able to learn the original model successfully despite the variability in both component means and mixing proportions of the mixture model.

## 4   A model for fMRI activation surfaces

We now apply the general framework of the hierarchical DP with random effects to the problem of detecting and characterizing spatial activation patterns in fMRI brain images. Underlying our approach is an assumption that there is an unobserved true spatial activation pattern in a subject's brain given a particular stimulus and that multiple activation images for this individual collected over different fMRI sessions are realizations of the true activation image, with variability in the activation pattern due to various sources. Our goal is to infer the unknown true activation from multiple such activation images.

We model each activation image using a mixture of experts model, with a component expert assigned to each local activation cluster (Rasmussen and Ghahramani, 2002). By introducing a hierarchical DP into this model we allow activation clusters to be shared across images, inferring the number of such clusters from the data. In addition, the random effects component can be incorporated to allow activation centers to be slightly shifted in terms of pixel locations or in terms of peak intensity. These types of variation are common in multi-image fMRI experiments, due to a variety of factors such as head motion, variation in the physiological and cognitive states of the subject. In what follows below we will focus on 2-dimensional "slices" rather than 3-dimensional voxel images—in principle the same type of model could be developed for the 3-dimensional case.

We briefly discuss the mixture of experts model below (Kim et al., 2006). Assuming the $\beta$ values $y_i, i = 1, \ldots, N$ are conditionally independent of each other given the voxel position $\mathbf{x}_i = (x_{i1}, x_{i2})$ and the model parameters, we model the activation $y_i$ at voxel $\mathbf{x}_i$ as a mixture of experts:

$$p(y_i|\mathbf{x}_i, \theta) = \sum_{c \in \mathscr{C}} p(y_i|c, \mathbf{x}_i) P(c|\mathbf{x}_i), \tag{7}$$

where $\mathscr{C} = \{c_{bg}, c_m, m = 1, \ldots, M-1\}$ is a set of $M$ expert component labels for background $c_{bg}$ and $M-1$ activation components $c_m$'s. The first term on the right hand side of Equation (7) defines the expert for a given component. We model the expert for an activation component as a

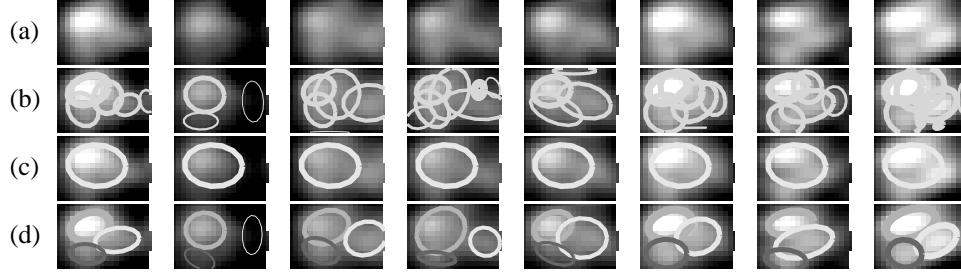

Figure 4: Results from eight runs for subject 2 at Stanford. (a) Raw images for a cross section of right precentral gyrus and surrounding area. Activation components estimated from the images using (b) DP mixtures, (c) hierarchical DPs, and (d) hierarchical DP with random effects.

Gaussian-shaped surface centered at $\mathbf{b}_m$ with width $\Sigma_m$ and height $h_m$ as follows.

$$y_i = h_m\exp\left(-(\mathbf{x}_i - \mathbf{b}_m)'(\Sigma_m)^{-1}(\mathbf{x}_i - \mathbf{b}_m)\right) + \varepsilon, \tag{8}$$

where $\varepsilon$ is an additive noise term distributed as $\mathcal{N}(0, \sigma_{\text{act}}^2)$. The background component is modeled as $y_i = \mu + \varepsilon$, having a constant activation level $\mu$ with additive noise distributed as $\mathcal{N}(0, \sigma_{\text{bg}}^2)$.

The second term in Equation (7) is known as a gate function in the mixture of experts framework—it decides which expert should be used to make a prediction for the activation level at position $\mathbf{x}_i$. Using Bayes' rule we write this term as $P(c|\mathbf{x}_i) = p(\mathbf{x}_i|c)\pi_c/(\sum_{c \in \mathscr{C}} p(\mathbf{x}_i|c)\pi_c)$, where $\pi_c$ is a class prior probability $P(c)$. $p(\mathbf{x}_i|c)$ is defined as follows. For activation components, $p(\mathbf{x}_i|c_m)$ is a normal density with mean $\mathbf{b}_m$ and covariance $\Sigma_m$. $\mathbf{b}_m$ and $\Sigma_m$ are shared with the Gaussian surface model for experts in Equation (8). This implies that the probability of activating the $m$th expert is highest at the center of the activation and gradually decays as $\mathbf{x}_i$ moves away from the center. $p(\mathbf{x}_i|c_{bg})$ for the background component is modeled as having a uniform distribution of $1/N$ for all positions in the brain. If $\mathbf{x}_i$ is not close to the center of any activations, the gate function selects the background expert for the voxel.

We place a hierarchical DP prior on $\pi_c$, and let the location parameters $\mathbf{b}_m$ and the height parameters $h_m$ vary in individual images according to a Normal prior distribution with a variance $\mathbf{\Psi}_{\mathbf{b}_m}$ and $\psi_{h_m}^2$ using a random effects model. We define prior distributions for $\mathbf{\Psi}_{\mathbf{b}_m}$ and $\psi_{h_m}^2$ as a half normal distribution with a 0 mean and a variance as suggested by Gelman (2006). Since the surface model for the activation component is a highly non-linear model, without conjugate prior distributions it is not possible to evaluate the integrals in Equations (5b)-(5c) and (6) analytically in the sampling algorithm. We rely on an approximation of the integrals by sampling new values for $\mathbf{b}_m$ and $h_m$ from their priors and new values for image-specific random effects parameters from $\mathcal{N}(\mathbf{b}_m, \mathbf{\Psi}_{\mathbf{b}_m})$ and $\mathcal{N}(h_m, \psi_{h_m}^2)$ and evaluating the likelihood of the data given these new values for the unknown parameters.

## 5 Experimental results on fMRI data

We demonstrate the performance of the model and inference algorithm described above by using fMRI data collected from three subjects (referred to as Subjects 1, 2 and 3) performing the same sensorimotor task at two different fMRI scanners (Stanford and Duke). Each subject was scanned during eight separate fMRI experiments ("runs") and for each run a $\beta$-map (a voxel image that summarizes the brain activation) was produced using standard fMRI preprocessing.

In this experiment we analyze a 2D cross-section of the right precentral gyrus brain region, a region that is known to be activated by this sensorimotor task. We fit our model to each set of eight $\beta$-maps for each of the subjects at each scanner, and compare the results from the models obtained from the hierarchical DP without random effects. We also fit standard DP mixtures to individual images as a baseline, using Algorithm 7 from Neal (1998) to sample from the model. The concentration parameters for DP priors in all of the three models were given a prior distribution gamma(1.5, 1) and sampled from the posterior as described in Teh et al.(2006). For all of the models the MCMC sampling algorithm was run for 3000 iterations.

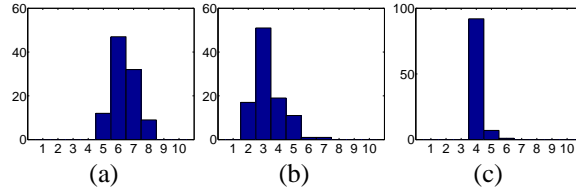

Figure 5: Histogram of the number of components over the last 1000 iterations (Subject 2 at Stanford). (a) DP mixture, (b) hierarchical DP, and (c) hierarchical DP with random effects.

| Scanner | Subject | Hierarchical DP | | Hierarchical DP with random effects | |
|---|---|---|---|---|---|
| | | Avg. logP | Standard deviation | Avg. logP | Standard deviation |
| Stanford | Subject 1 | -1142.6 | 21.8 | -1085.3 | 12.6 |
| | Subject 2 | -1260.9 | 32.1 | -1082.8 | 28.7 |
| | Subject 3 | -1084.1 | 11.3 | -1040.9 | 13.5 |
| Duke | Subject 1 | -1154.9 | 12.5 | -1166.9 | 13.1 |
| | Subject 2 | -677.9 | 12.2 | -559.9 | 15.8 |
| | Subject 3 | -1175.6 | 13.6 | -1086.8 | 13.2 |

Table 2: Predictive logP scores of test images averaged over eight cross-validation runs. The simulation errors are shown as standard deviations.

Figure 4(a) shows $\beta$-maps from eight fMRI runs of Subject 2 at Stanford. From the eight images one can see three primary activation bumps, subsets of which appear in different images with variability in location and intensity. Figures 4 (b)-(d) each show a sample from the model learned on the data in Figure 4(a), where Figure 4(b) is for DP mixtures, Figure 4(c) for hierarchical DPs, and Figure 4(d) for hierarchical DPs with random effects. The sampled activation components are overlaid as ellipses using one standard deviation of the width parameters $\Sigma_m$. The thickness of ellipses indicates the estimated height $h_m$ of the bump. In Figures 4(b) and (c) ellipses for activation components shared across images are drawn with the same color.

The DPs shown in Figure 4(b) seem to overfit with many bumps and show a relatively poor generalization capability because the model cannot borrow strength from other similar images. The hierarchical DP in Figure 4(c) is not flexible enough to account for bumps that are shared across images but that have variability in their parameters. By using one fixed set of component parameters shared across images, the hierarchical DPs are too constrained and are unable to detect the more subtle features of individual images. The random effects model finds the three main bumps and a few more bumps with lower intensity for the background. Thus, in terms of generalization, the model with random effects provides a good trade-off between the relatively unconstrained DP mixtures and overly-constrained hierarchical DPs. Histograms of the number of components (every 10 samples over the last 1000 iterations) for the three different models are shown in Figure 5.

We also perform a leave-one-image-out cross-validation to compare the predictive performance of hierarchical DPs and our proposed model. For each subject at each scanner we fit a model from seven images and compute the predictive likelihood of the remaining one image. The predictive scores and simulation errors (standard deviations) averaged over eight cross-validation runs for both models are shown in Table 2. In all of the subjects except for Subject 1 at Duke, the proposed model shows a significant improvement over hierarchical DPs. For Subject 1 at Duke, the hierarchical DP gives a slightly better result but the difference in scores is not significant relative to the simulation error.

Figure 6 shows the difference in the way the hierarchical DP and our proposed model fit the data in one cross-validation run for Subject 1 at Duke as shown in Figure 6(a). The hierarchical DP in Figure 6(b) models the common bump with varying intensity in the middle of each image as a mixture of two components—one for the bump in the first two images with relatively high intensity and another for the same bump in the rest of the images with lower intensity. Our proposed model recovers the correspondence in the bumps with different intensity across images as shown in Figure 6(c).

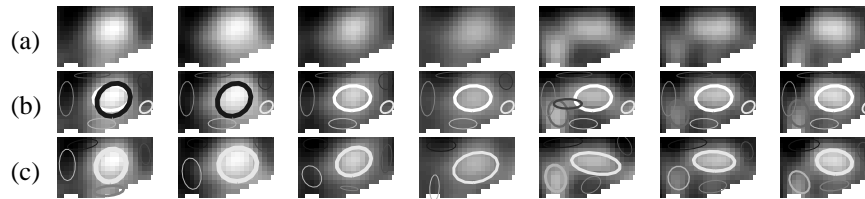

Figure 6: Results from one cross-validation run for subject 1 at Duke. (a) Raw images for a cross section of right precentral gyrus and surrounding area. Activation components estimated from the images are shown in (b) for hierarchical DPs, and in (c) for hierarchical DP with random effects.

# 6   Conclusions

In this paper we proposed a hierarchical DP model with random effects that allows each group (or image) to have group-level mixture component parameters as well as group-level mixing proportions. Using fMRI brain activation images we demonstrated that our model can capture components shared across multiple groups with individual-level variation. In addition, we showed that our model is able to estimate the number of components more reliably due to the additional flexibility in the model compared to DP mixtures and hierarchical DPs. Possible future directions for this work include extensions to modeling differences between labeled groups of individuals, e.g., in studies of controls and patients for a particular disorder.

## Acknowledgments

We would like to thank Hal Stern for useful discussions. We acknowledge the support of the following grants: the Functional Imaging Research in Schizophrenia Testbed, Biomedical Informatics Research Network (FIRST BIRN; 1 U24 RR021992, www.nbirn.net); the Transdisciplinary Imaging Genetics Center (P20RR020837-01); and the National Alliance for Medical Image Computing (NAMIC; Grant U54 EB005149), funded by the National Institutes of Health through the NIH Roadmap for Medical Research. Author PS was also supported in part by the National Science Foundation under awards number IIS-0431085 and number SCI-0225642.

## References

Gelman, A., Carlin, J., Stern, H. & Rubin, D. (2004) *Bayesian Data Analysis*, New York: Chapman & Hall/CRC.

Gelman, A. (2006). Prior distribution for variance parameters in hierarchical models. *Bayesian Analysis*, 1(3):515–533.

Kim, S., Smyth, P., & Stern, H. (2006). A nonparametric Bayesian approach to detecting spatial activation patterns in fMRI data. *Proceedings of the 9th International Conference on Medical Image Computing and Computer Assisted Intervention,* vol. 2, pp.217–224.

Neal, R.M. (1998) Markov chain sampling methods for Dirichlet process mixture models. Technical Report 4915, Department of Statistics, University of Toronto.

Penny, W. & Friston, K. (2003) Mixtures of general linear models for functional neuroimaging. *IEEE Transactions on Medical Imaging*, **22**(4):504–514.

Rasmussen, C.E. (2000) The infinite Gaussian mixture model. *Advances in Neural Information Processing Systems 12*, pp. 554–560. MIT Press.

Rasmussen, C.E. & Ghahramani, Z. (2002) Infinite mixtures of Gaussian process experts. *Advances in Neural Information Processing Systems 14*, pp. 881–888. MIT Press.

Sethuraman, J. (1994) A constructive definition of Dirichlet priors. *Statistica Sinica*, 4:639–650.

Sudderth, E., Torralba, A., Freeman, W. & Willsky, A. (2005). Describing visual scenes using transformed Dirichlet Processes. *Advances in Neural Information Processing Systems 18,* pp. 1297–1304. MIT Press.

Teh, Y.W., Jordan, M.I., Beal, M.J. & Blei, D.M. (2006). Hierarchical Dirichlet processes. *Journal of American Statistical Association, To appear.*
